# Bayesian modelling of fMRI time series

**Pedro A. d. F. R. Højen-Sørensen, Lars K. Hansen** and **Carl Edward Rasmussen**
Department of Mathematical Modelling, Building 321
Technical University of Denmark
DK-2800 Lyngby, Denmark
`phs,lkhansen,carl@imm.dtu.dk`

## Abstract

We present a Hidden Markov Model (HMM) for inferring the hidden psychological state (or neural activity) during single trial fMRI activation experiments with blocked task paradigms. Inference is based on Bayesian methodology, using a combination of analytical and a variety of Markov Chain Monte Carlo (MCMC) sampling techniques. The advantage of this method is that detection of short time learning effects between repeated trials is possible since inference is based only on single trial experiments.

## 1 Introduction

Functional magnetic resonance imaging (fMRI) is a non-invasive technique that enables indirect measures of neuronal activity in the working human brain. The most common fMRI technique is based on an image contrast induced by temporal shifts in the relative concentration of oxyhemoglobin and deoxyhemoglobin (BOLD contrast). Since neuronal activation leads to an increased blood flow, the so-called *hemodynamic response*, the measured fMRI signal reflects neuronal activity. Hence, when analyzing the BOLD signal there are two unknown factors to consider; the task dependent neuronal activation and the hemodynamic response. Bandettini et al. [1993] analyzed the correlation between a binary reference function (representing the stimulus/task sequence) and the BOLD signal. In the following we will also make reference to the binary representation of the task as the *paradigm*. Lange and Zeger [1997] discuss a parameterized hemodynamic response adapted by a least squares procedure. Multivariate strategies have been pursued in [Worsley et al. 1997, Hansen et al. 1999]. Several explorative strategies have been proposed for finding spatio-temporal activation patterns without explicit reference to the activation paradigm. McKeown et al. [1998] used independent component analysis and found several types of activations including components with "transient task related" response, i.e., responses that could not simply be accounted for by the paradigm. The model presented in this paper draws on the experimental observation that the basic coupling between the net neural activity and hemodynamic response is roughly linear while the relation between neuronal response and stimulus/task parameters is often nonlinear [Dale 1997]. We will represent the neuronal activity (integrated over the voxel and sampling time interval) by a binary signal while we will represent the hemodynamic response as a linear filter of unknown form and temporal extent.

## 2 A Bayesian model of fMRI time series

Let $s = \{s_t : t = 0, \ldots, T - 1\}$ be a hidden sequence of binary state variables $s_t \in \{0, 1\}$, representing the state of a single voxel over time; the time variable, $t$, indexes the sequence of fMRI scans. Hence, $s_t$ is a binary representation of the neural state. The hidden sequence is governed by a symmetric first order Hidden Markov Model (HMM) with transition probability $\alpha = P(S_{t+1} = j | S_t = j)$. We expect the activation to mimic the blocked structure of the experimental paradigm so for this reason we restrict $\alpha$ to be larger than one half. The *predicted signal* (noiseless signal) is given by $y_t = h * s + \theta_0 + \theta_1 t$, where $*$ denotes the linear convolution and $h$ is the impulse response of a linear system of order $M_f$. The dc off-set and linear trend which are typically seen in fMRI time series are given by $\theta_0$ and $\theta_1$, respectively. Finally, it is assumed that the observable is given by $z_t = y_t + \varepsilon_t$, where $\varepsilon_t$ is iid. Gaussian noise with variance $\sigma_n^2$. The generative model considered is therefore given by:

$$
\begin{aligned}
p(s_t | s_{t-1}, \alpha) &= \alpha \delta_{s_t, s_{t-1}} + (1 - \alpha)(1 - \delta_{s_t, s_{t-1}}), \\
p(z | s, \sigma_n, \theta, M_f) &\sim \mathcal{N}(y, \sigma_n^2 I), \text{ where } y = \{y_t\} = H_s \theta, \text{ and } z = \{z_t\}.
\end{aligned} \qquad (*)
$$

Furthermore, $\delta_{s_t, s_{t-1}}$ is the usual Kronecker delta and $H_s = [\mathbf{1}, \xi, \gamma_0 s, \gamma_1 s, \ldots, \gamma_{M_f-1} s]$, where $\mathbf{1} = (1, \ldots 1)'$, $\xi = (1, \ldots, T)'/T$ and $\gamma_i$ is a $i$-step shift operator, that is $\gamma_i s = (0, \ldots, 0, s_0, s_1, \ldots, s_{T-1-i})'$. The linear parameters are collected in $\theta = (\theta_0, \theta_1, h)'$.

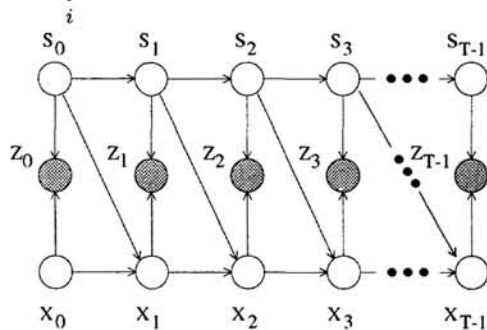

The graphical model. The hidden states $X_t = (S_{t-1}, S_{t-2}, \ldots, S_{t-(M_f-1)})$ have been introduced to make the model first order.

## 3 Analytic integration and Monte Carlo sampling

In this section we introduce priors over the model parameters and show how inference may be performed. The filter coefficients and noise parameters may be handled analytically, whereas the remaining parameters are treated using sampling procedures (a combination of Gibbs and Metropolis sampling). Like in the previous section explicit reference to the filter order $M_f$ may be omitted to ease the notation.

The dc off-set $\theta_0$ and the linear trend $\theta_1$ are given (improper) uniform priors. The filter coefficients are given priors that are uniform on an interval of length $\beta$ independently for each coefficient:

$$
p(h | M_f) = \begin{cases} \beta^{-M_f} & \text{for } |h_i| < \frac{\beta}{2}, \quad 0 \le i \le M_f - 1 \\ 0 & \text{otherwise} \end{cases}
$$

Assuming that all the values of $\theta$ for which the associated likelihood has non-vanishing contributions lie inside the box where the prior for $\theta$ has support, we may integrate out the filter coefficients via a Gaussian integral:

$$
p(z | \sigma_n, s, M_f) = \int p(z | \theta, \sigma_n, s, M_f) p(\theta | M_f) d\theta = \frac{(2\pi \sigma_n^2)^{\frac{M_f - T}{2} + 1}}{\beta^{M_f} \sqrt{|H_s' H_s|}} \exp\left(-\frac{z'z - \hat{y}_s' \hat{y}_s}{2\sigma_n^2}\right).
$$

We have here defined the *mean filter*, $\hat{\theta}_s = (H'_s H_s)^{-1} H_s z$ and *mean predicted signal*, $\hat{y}_s = H_s \hat{\theta}_s$, for given state and filter length. We set the interval-length, $\beta$ to be 4 times the standard deviation of the observed signal $z$. This is done, since the response from the filter should be able to model the signal, for which it is thought to need an interval of plus/minus two standard deviations.

We now proceed to integrate over the noise parameter; using the (improper) non-informative Jeffreys prior, $p(\sigma_n) \propto \sigma_n^{-1}$, we get a Gamma integral:

$$p(z|s, M_f) = \int p(z|\sigma_n, s, M_f) p(\sigma_n) d\sigma_n = \frac{1}{2}\Gamma\left(\frac{T - M_f}{2} - 1\right) \frac{\left(\pi(z'z - \hat{y}'_s \hat{y}_s)\right)^{\frac{M_f - T}{2} + 1}}{\beta^{M_f}\sqrt{|H'_s H_s|}}.$$

The remaining variables cannot be handled analytically, and will be treated using various forms of sampling as described in the following sections.

### 3.1 Gibbs and Metropolis updates of the state sequence

We use a flat prior on the states, $p(s_t = 0) = p(s_t = 1)$, together with the first order Markov property for the hidden states and Bayes' rule to get the conditional posterior for the individual states:

$$p(s_t = j|s \backslash s_t, \alpha, M_f) \quad \propto \quad p(s_t = j|s_{t-1}, \alpha)p(s_{t+1}|s_t = j, \alpha)p(z|s, M_f).$$

These probabilities may (in normalized form) be used to implement Gibbs updates for the hidden state variables, updating one variable at a time and sweeping through all variables. However, it turns out that there are significant correlations between states which makes it difficult for the Markov Chain to move around in the hidden state-space using only Gibbs sampling (where a single state is updated at a time). To improve the situation we also perform global state updates, consisting of proposing to move the entire state sequence one step forward or backward (the direction being chosen at random) and accepting the proposed state using the Metropolis acceptance procedure. The proposed movements are made using periodic boundary conditions. The Gibbs sweep is computationally involved, since it requires computation of several matrix expressions for every state-variable.

### 3.2 Adaptive Rejection Sampling for the transition probability

The likelihood for the transition probability $\alpha$ is derived from the Hidden Markov Model:

$$p(s|\alpha) = p(s_0) \prod_{t=1}^{T-1} p(s_t|s_{t-1}, \alpha) = \frac{1}{2}\alpha^{E(s)}(1 - \alpha)^{T-1-E(s)},$$

where $E(s) = \sum_{t=1}^{T-1} \delta_{s_t, s_{t-1}}$ is the number of neighboring states in $s$ with identical values. The prior on the transition probabilities is uniform, but restricted to be larger than one half, since we expect the activation to mimic the blocked structure of the experimental paradigm. It is readily seen that $p(\alpha|s) \propto p(s|\alpha)$, $\alpha \in [\frac{1}{2}, 1]$ is log-concave. Hence, we may use the Adaptive Rejection Sampling algorithm [Gilks and Wild, 1992] to sample from the distribution for the transition probability.

### 3.3 Metropolis updates for the filter length

In practical applications using real fMRI data, we do typically not know the necessary length of the filter. The problem of finding the "right" model order is difficult and has received a lot of attention. Here, we let the Markov Chain sample over different filter lengths, effectively integrating out the filter-length rather than trying to optimize it. Although the

value of $M_f$ determines the dimensionality of the parameter space, we do not need to use specialized sampling methodology (such as Reversible Jump MCMC [Green, 1995]), since those parameters are handled analytically in our model. We put a flat (improper) prior on $M_f$ and propose new filter lengths using a Gaussian proposal centered on the current value, with a standard deviation of 3 (non-positive proposed orders are rejected). This choice of the standard deviation only effects the mixing rate of the Markov chain and does not have any influence on the stationary distribution. The proposed values are accepted using the Metropolis algorithm, using $p(M_f|s,y) \propto p(y|s,M_f)$.

### 3.4 The posterior mean and uncertainty of the predicted signal

Since $\theta$ has a flat prior the conditional probability for the filter coefficients is proportional to the likelihood $p(z|\theta, \cdot)$ and by $(*)$ we get:

$$p(\theta|z,s,\sigma_{\mathrm{n}},M_f) \sim \mathcal{N}(D_s z, \sigma_{\mathrm{n}}^2 D_s D_s'), \quad D_s = (H_s' H_s)^{-1} H_s' \,.$$

The posterior mean of the predicted signal, $\hat{y}$, is then readily computed as:

$$\hat{y} = \langle y_{\theta,\sigma_{\mathrm{n}},s,M_f} \rangle_{\theta,\sigma_{\mathrm{n}},s,M_f} = \langle \hat{y}_s \rangle_{s,M_f} = \langle H_s \hat{\theta}_s \rangle_{s,M_f} = \langle F_s \rangle_{s,M_f} z \,,$$

where $F_s = H_s D_s$. Here, the average over $\theta$ and $\sigma_{\mathrm{n}}$ is done analytically, and the average over the state and filter length is done using Monte Carlo. The uncertainty in the posterior, can also be estimated partly by analytical averaging, and partly Monte Carlo:

$$\begin{aligned}
\Sigma_y &= \langle (y_{\theta,\sigma_{\mathrm{n}},s,M_f} - \hat{y})(y_{\theta,\sigma_{\mathrm{n}},s,M_f} - \hat{y})' \rangle_{\theta,\sigma_{\mathrm{n}},s,M_f} \\
&= \frac{1}{T - M_f - 2} \langle (z'z - \hat{y}_s'\hat{y}_s) F_s F_s' \rangle_{s,M_f} + \langle F_s zz' F_s' \rangle_{s,M_f} - \hat{y}\hat{y}' \,.
\end{aligned}$$

## 4  Example: synthetic data

In order to test the model, we first present some results on a synthetic data set. A signal $z$ of length 100 is generated using a $M_f = 10$ order filter, and a hidden state sequence $s$ consisting of two activation bursts (indicated by dotted bars in figure 1 top left). In this example, the hidden sequence is actually not generated from the generative model $(*)$; however, it still exhibits the kind of block structure that we wish to be able to recover. The model is run for 10000 iterations, which is sufficient to generate 500 approximately independent samples from the posterior; figure 2 (right) shows the auto-covariance for $M_f$ as a function of the iteration lag. It is thought that changes in $M_f$ are indicative of correlation time of the overall system.

The correlation plot for the hidden states (figure 2, left) shows that the state activation onset correlates strongly with the second onset and negatively with the end of the activation (and vice versa). This indicates that the Metropolis updates described in section 3.1 may indeed be effective. Notice also that the very strong correlation among state variables does not strongly carry over to the predicted signal (figure 1, bottom right).

To verify that the model can reasonably recover the parameters used to generate the data, posterior samples from some of the model variables are shown in figure 3. For all these parameters the posterior density is large around the correct values. Notice, that there in the original model $(*)$ is an indeterminacy in the simultaneous inference of the state sequence and the filter parameters (but no indeterminacy in the predicted signal); for example, the same signal is predicted by shifting the state sequence backward in time and introducing leading zero filter coefficients. However, the Bayesian methodology breaks this symmetry by penalizing complex models.

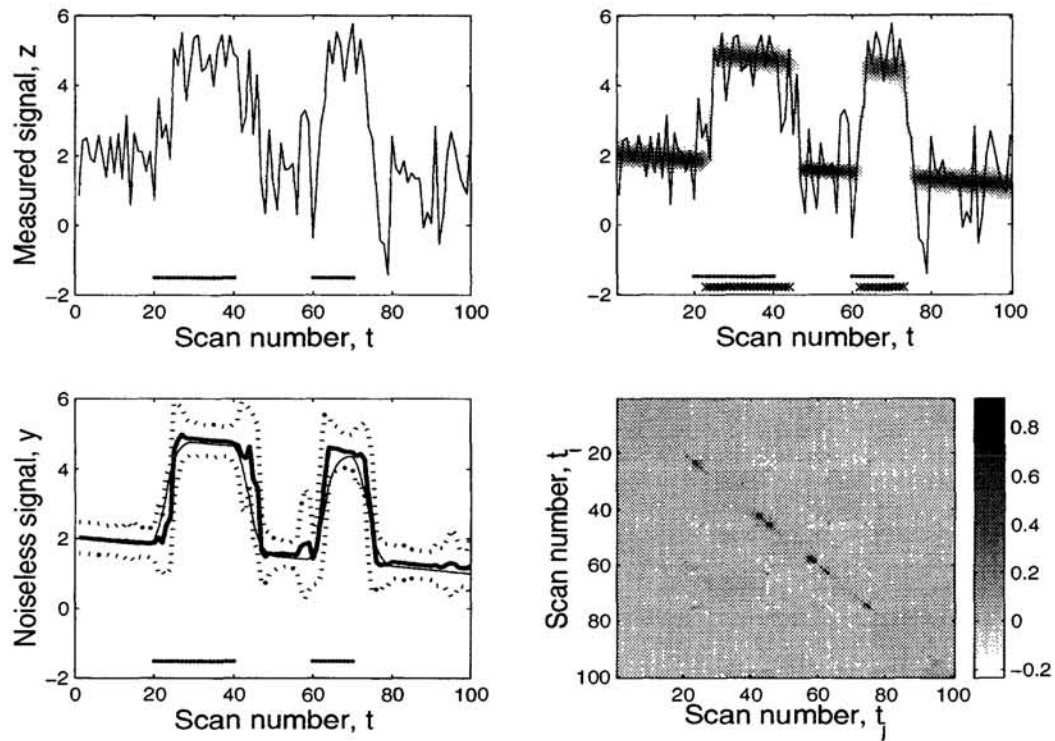

Figure 1: Experiments with synthetic data. Top left, the measured response from a voxel is plotted for 100 consecutive scans. In the bottom left, the underlying signal is seen in thin, together with the posterior mean, $\hat{y}$ (thick), and two std. dev. error-bars in dotted. Top right, the posterior probabilities are shown as a grey level, for each scan. The true activated instances are indicated by the dotted bars and the pseudo MAP estimate of the activation sequence is given by the crossed bars. Bottom right, shows the posterior uncertainty $\Sigma_y$.

The posterior mean and the two standard deviations are plotted in figure 1 bottom left. Notice, however, that the distribution of $y$ is not Gaussian, but rather a mixture of Gaussians, and is not necessarily well characterized by mean and variance alone. In figure 1 (top left), the distribution of $y_t$ is visualized using grey-scale to represent density.

## 5    Simulations on real fMRI data and discussion

In figure 4 the model has been applied to two measurements in the same voxel in visual cortex. The fMRI scans were acquired every 330 ms. The experimental paradigm consisted of 30 scans of rest followed by 30 scans of activation and 40 rest. Visual activation consisted of a flashing (8 Hz) annular checkerboard pattern. The model readily identifies the activation burst of somewhat longer duration than the visual stimulus and delayed around 2 seconds. The delay is in part caused by the delay in the hemodynamic response.

These results show that the integration procedure works in spite of the very limited data at hand. In figure 4 (top) the posterior model size suggests that (at least) two competing models can explain the signal from this trial. One of these models explains the measured signal as a simple square wave function which seems reasonable by considering the signal. Conversely, figure 4 (bottom), suggests that the signal from the second trial can not be explained by a simple model. This too, seems reasonable because of the long signal raise interval suggested in the signal.

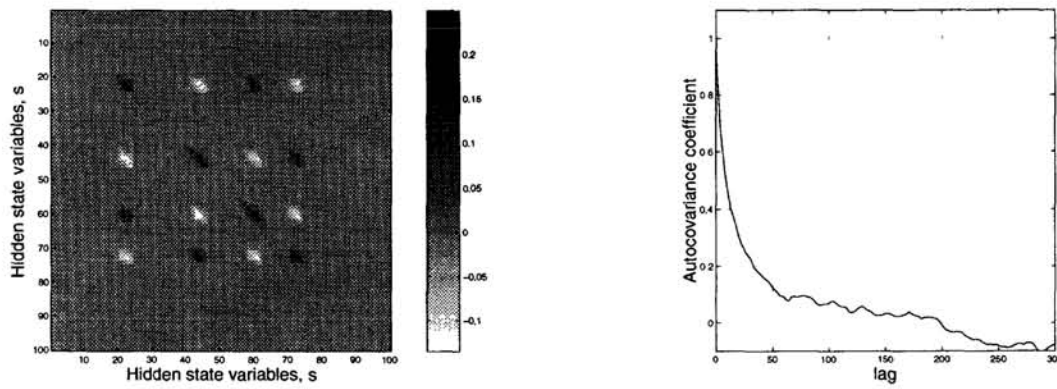

Figure 2: The covariance of the hidden states based on a long run of the model is shown to the left. Notice, that the states around the front (back) of the activity "bumps" are highly (anti-) correlated. Right: The auto-covariance for the filter length $M_f$ as a function of the lag time in iterations. The correlation length is about 20, computed as the sum of auto-covariance coefficients from lag $-400$ to $400$.

Since the posterior distribution of the filter length is very broad it is questionable whether an optimization based procedure such as maximum likelihood estimation would be able to make useful inference in this case were data is very limited. Also, it is not obvious how one may use cross-validation in this setting. One might expect such optimization based strategies to get trapped in suboptimal solutions. This, of course, remains to be investigated.

## 6  Conclusion

We have presented a model for voxel based explorative data analysis of single trial fMRI signals during blocked task activation studies. The model is founded on the experimental observation that the basic coupling between the net neural activity and hemodynamic response is roughly linear. The preliminary investigation reported here are encouraging in that the model reliably detects reasonable hidden states from the very noisy fMRI data.

One drawback of this method is that the Gibbs sampling step is computational expensive. To improve on this step one could make use of the large class of variational/mean field methods known from the graphical models literature. Finally, current work is in progress for generalizing the model to multiple voxels, including spatial correlation due to e.g. spill-over effects.

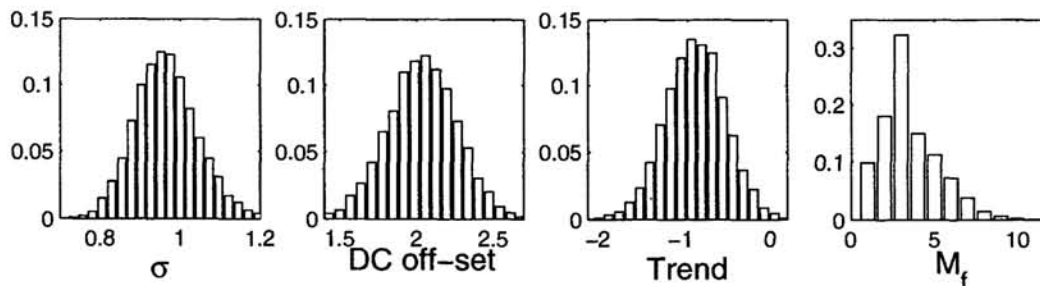

Figure 3: Posterior distributions of various model parameters. The parameters used to generate the data are: $\sigma = 1.0$, DC off-set $= 2$, trend $= -1$ and filter order $M_f = 10$.

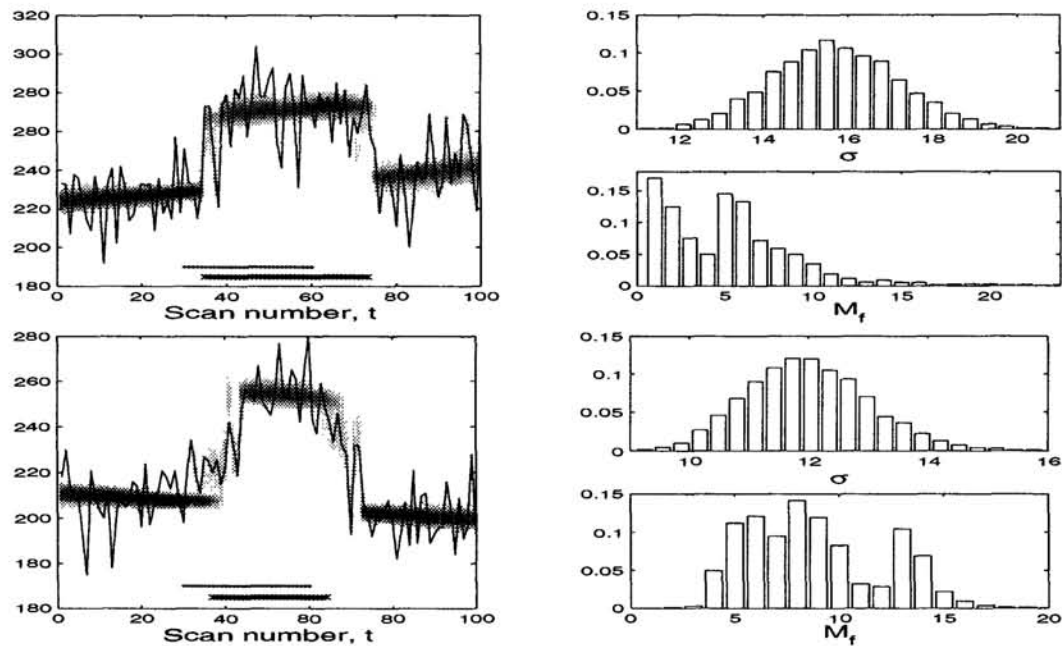

Figure 4: Analysis of two experimental trials of the same voxel in visual cortex. The left hand plot shows the posterior inferred signal distribution superimposed by the measured signal. The dotted bar indicates the experimental paradigm and the crossed bar indicates the pseudo MAP estimate of the neural activity. To the right the posterior noise level and inferred filter length are displayed.

## Acknowledgments

Thanks to Egill Rostrup for providing the fMRI data. This work is funded by the Danish Research Councils through the Computational Neural Network Center (CONNECT) and the THOR Center for Neuroinformatics.

## References

Bandettini, P. A. (1993). Processing strategies for time-course data sets in functional MRI of the human brain *Magnetic Resonance in Medicine 30*, 161–173.

Dale, A. M. and R. L. Buckner (1997). Selective Averaging of Individual Trials Using fMRI. *NeuroImage 5*, Abstract S47.

Green, P. J. (1995). Reversible jump Markov chain Monte Carlo computation and Bayesian model determination. *Biometrika 82*, 711–732.

Gilks, W. R. and P. Wild (1992). Adaptive rejection sampling for Gibbs sampling. *Applied Statistics 41*, 337–348.

Hansen, L. K. et al. (1999). Generalizable Patterns in Neuroimaging: How Many Principal Components? *NeuroImage*, to appear.

Lange, N. and S. L. Zeger (1997). Non-linear Fourier time series analysis for human brain mapping by functional magnetic resonance imaging. *Journal of the Royal Statistical Society - Series C Applied Statistics 46*, 1–30.

McKeown, M. J. et al. (1998). Spatially independent activity patterns in functional magnetic resonance imaging data during the stroop color-naming task. *Proc. Natl. Acad. Sci. USA. 95*, 803–810.

Worsley, K. J. et al. (1997). Characterizing the Response of PET and fMRI Data Using Multivariate Linear Models (MLM). *NeuroImage 6*, 305–319.